# Analog Soft-Pattern-Matching Classifier using Floating-Gate MOS Technology

**Toshihiko YAMASAKI and Tadashi SHIBATA***

Department of Electronic Engineering, School of Engineering
*Department of Frontier Informatics, School of Frontier Science
The University of Tokyo
7-3-1 Hongo, Bunkyo-ku, Tokyo, 113-8656, Japan
*yamasaki@if.t.u-tokyo.ac.jp, shibata@ee.t.u-tokyo.ac.jp*

## Abstract

A flexible pattern-matching analog classifier is presented in conjunction with a robust image representation algorithm called Principal Axes Projection (PAP). In the circuit, the functional form of matching is configurable in terms of the peak position, the peak height and the sharpness of the similarity evaluation. The test chip was fabricated in a 0.6- $\mu$m CMOS technology and successfully applied to hand-written pattern recognition and medical radiograph analysis using PAP as a feature extraction pre-processing step for robust image coding. The separation and classification of overlapping patterns is also experimentally demonstrated.

## 1 Introduction

Pattern classification using template matching techniques is a powerful tool in implementing human-like intelligent systems. However, the processing is computationally very expensive, consuming a lot of CPU time when implemented as software running on general-purpose computers. Therefore, software approaches are not practical for real-time applications. For systems working in mobile environment, in particular, they are not realistic because the memory and computational resources are severely limited. The development of analog VLSI chips having a fully parallel template matching architecture [1,2] would be a promising solution in such applications because they offer an opportunity of low-power operation as well as very compact implementation.

In order to build a real human-like intelligent system, however, not only the pattern representation algorithm but also the matching hardware itself needs to be made flexible and robust in carrying out the pattern matching task. First of all, two-dimensional patterns need to be represented by feature vectors having substantially reduced dimensions, while at the same time preserving the human perception of similarity among patterns in the vector space mapping. For this purpose, an image representation algorithm called Principal Axes Projection (PAP) has been de-

veloped[3] and its robust nature in pattern recognition has been demonstrated in the applications to medical radiograph analysis [3] and hand-written digits recognition [4]. However, the demonstration so far was only carried out by computer simulation.

Regarding the matching hardware, high-flexibility analog template matching circuits have been developed for PAP vector representation. The circuits are flexible in a sense that the matching criteria (the weight to elements, the strictness in matching) are configurable. In Ref. [5], the fundamental characteristics of the building block circuits were presented, and their application to simple hand-written digits was presented in Ref. [6]. The purpose of this paper is to demonstrate the robust nature of the hardware matching system by experiments. The classification of simple hand-written patterns and the cephalometric landmark identification in gray-scale medical radiographs have been carried out and successful results are presented. In addition, multiple overlapping patterns can be separated without utilizing a priori knowledge, which is one of the most difficult problems at present in artificial intelligence.

## 2 Image representation by PAP

PAP is a feature extraction technique using the edge information. The input image (64x64 pixels) is first subjected to pixel-by-pixel spatial filtering operations to detect edges in four directions: horizontal (HR); vertical (VR); +45 degrees (+45); and −45 degrees (-45). Each detected edge is represented by a binary flag and four edge maps are generated. The two-dimensional bit array in an edge map is reduced to a one-dimensional array of numerals by projection. The horizontal edge flags are accumulated in the horizontal direction and projected onto vertical axis. The vertical, +45-degree and −45-degree edge flags are similarly projected onto horizontal, -45-degree and +45-degree axes, respectively. Therefore the method is called "Principal Axes Projection (PAP)" [3,4]. Then each projection data set is series connected in the order of HR, +45, VR, -45 to form a feature vector. Neighboring four elements are averaged and merged to one element and a 64-dimensional vector is finally obtained. This vector representation very well preserves the human perception of similarity in the vector space. In the experiments below, we have further reduced the feature vector to 16 dimensions by merging each set of four neighboring elements into one, without any significant degradation in performance.

## 3 Circuit configurations

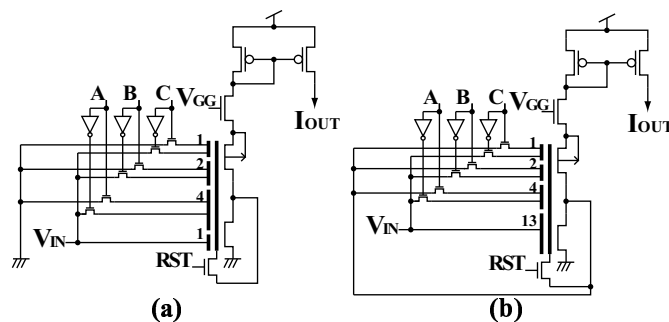

Figure 1: Schematic of vector element matching circuit: (a) pyramid (gain reduction) type; (b) plateau (feedback) type. The capacitor area ratio is indicated in the figure.

The basic functional form of the similarity evaluation is generated by the shortcut current flowing in a CMOS inverter as in Refs. [7,8,9]. However, their circuits were utilized to form radial basis functions and only the peak position was programmable. In our circuits, not only the peak position but also the peak height and the sharpness of the peak response shape are made configurable to realize flexible matching operations[5].

Two types of the element matching circuit are shown in Fig. 1. They evaluate the similarity between two vector elements. The result of the evaluation is given as an output current ($I_{OUT}$) from the pMOS current mirror. The peak position is temporarily memorized by auto-zeroing of the CMOS inverter. The common-gate transistor with $V_{GG}$ stabilizes the voltage supply to the inverter. By controlling the gate bias $V_{GG}$, the peak height can be changed. This corresponds to multiplying a weight factor to the element. The sharpness of the functional form is taken as the strictness of the similarity evaluation. In the pyramid type circuit (Fig. 1(a)), the sharpness is controlled by the gain reduction in the input. In the plateau type (Fig. 1(b)), the output voltage of the inverter is fed back to input nodes and the sharpness changes in accordance with the amount of the feedback.

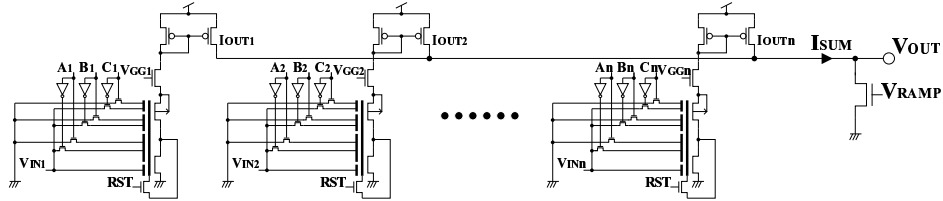

Figure 2: Schematic of n-dimensional vector matching circuit utilizing the pyramid type vector element circuits.

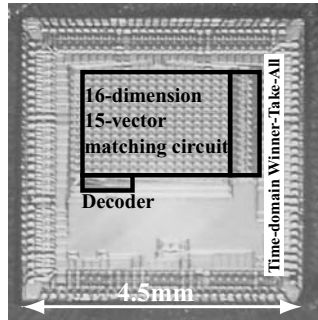

Figure 3: Photomicrograph of soft-pattern-matching classi fier circuit.

The total matching score between input and template vectors is obtained by taking the wired sum of all $I_{OUT}$'s from the element matching circuits as shown in Fig. 2. A multiplier circuit as utilized in Ref. [8] was eliminated because the radial basis function is not suitable for the template matching using PAP vectors. $I_{SUM}$, the sum of $I_{OUT}$'s, is then sunk through the nMOS with the $V_{RAMP}$ input. This forms a current comparator circuit, which compares $I_{SUM}$ and the sink current in the nMOS with $V_{RAMP}$. The $V_{OUT}$ nodes are connected to a time-domain Winner-Take-All circuit[9]. A common ramp down voltage is applied to the $V_{RAMP}$ nodes of all vector matching circuits. When $V_{RAMP}$ is ramped down from $V_{DD}$ to 0V, the vector matching circuit yielding the maximum $I_{SUM}$ firstly upsets and its output voltage ($V_{OUT}$) shows a 0-to-1 transition. The time-domain WTA circuit senses the first upsetting signal and memorizes the location in the open-loop OR-tree architecture [10]. In this manner, the maximum-likelihood template vector is easily identified.

The circuits were designed and fabricated in a 0.6-    µm double-poly triple-metal CMOS technology. Fig. 3 shows the photomicrograph of a pattern classifier circuit for 16-dimensional vectors. It contains 15 vector matching circuits. One element matching circuit occupies the area of 150    µm x 110 µm. In the latest design, however, the area is reduced to 54    µm x 68 µm in the same technology by layout optimization. Further area reduction is anticipated by employing high-K dielectric films for capacitors since the capacitors occupy a large area. The full functioning of the chip was experimentally confirmed [6]. In the following experiments, the simple vector matching circuit in Fig. 2 was utilized to investigate the response from each template vector instead of just detecting the winner using the full chip.

## 4   Experimental results and discussion

### 4.1   Vector-element matching circuit

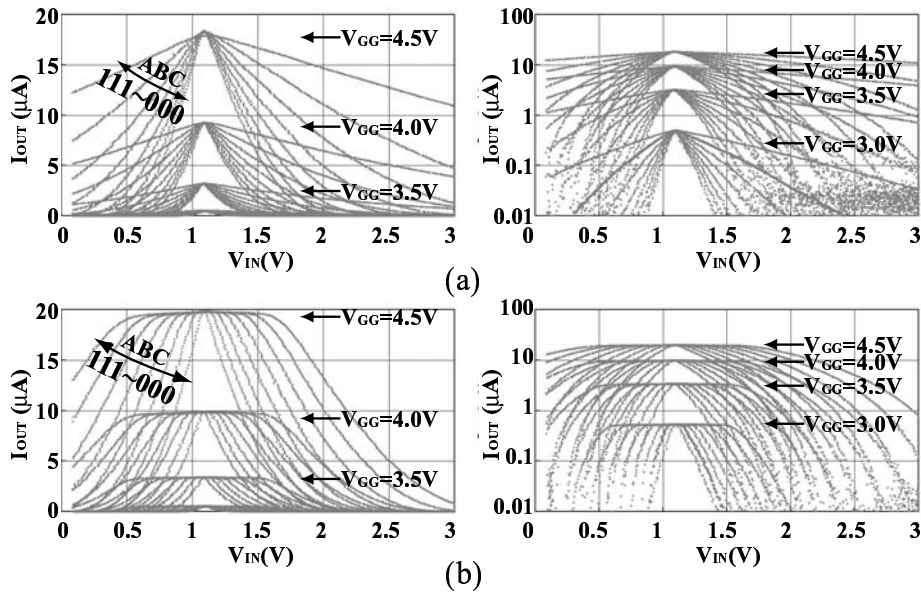

Figure 4: Measured characteristics: (a) pyramid type; (b) plateau type. V$_{GG}$ was varied from 3.0 V to 4.5 V, and control signals A~C from 000 to 111 for sharpness control.

Fig. 4 shows the measured characteristics of vector-element matching circuits in both linear and log plots. The peak position was set at 1.05 V by auto-zeroing. The peak height was altered by V$_{GG}$. Also, the operation mode was altered from the above-threshold region to the sub-threshold region by V$_{GG}$. In the plateau type circuit (Fig. 4(b)), I$_{OUT}$ becomes constant around the peak position and the flat region widens in proportion to the amount of feedback. This is because the inverter operates so as to keep the floating gate potential constant in the high-gain region of the inverter as in the case of virtual ground of an operational amplifier.

## 4.2 Matching of simple hand-written patterns

Fig. 5 demonstrates the matching results for the simple input patterns. 16 templates were stored in the matching circuit and several hand-written pattern vectors were presented to the circuit as inputs. A slight difference in the matching score is observed between the pyramid type and the plateau type, but the answers are correct for both types. Fig. 6 shows the effect of sharpness variation. As the sharpness gets steeper, all the scores decrease. However, the score ratios between the winner and loosers are increased, thus enhancing the winner discrimination margins. The matching results with varying operational regimes of the circuit are given in Fig. 7. The circuit functions properly even in the sub-threshold regime, demonstrating the opportunity of extremely low power operation.

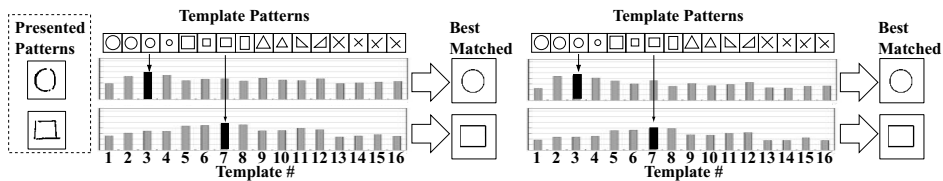

Figure 5: Result of simple pattern matching: (a) pyramid type (left) where gain reduction level was set with ABC=010; (b) plateau type (right) where feedback ratio was set with ABC=101.

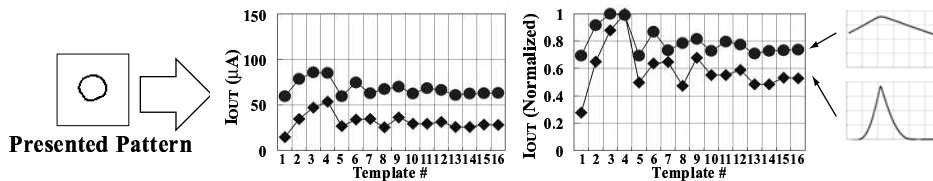

Figure 6: Effect of sharpness variation in the pyramid type with ABC=010.

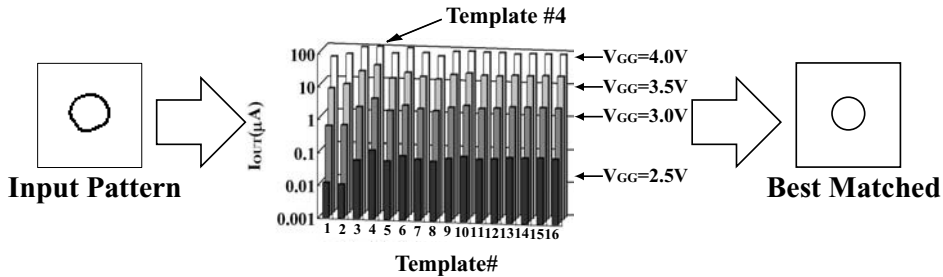

Figure 7: Matching results as a function of V$_{GG}$. Correct results are obtained in the sub-threshold regime as well as in the above-threshold regime (the pyramid type was utilized).

## 4.3 Application to gray-scale medical radiograph analysis

In Fig. 8, are presented the result of cephalometric landmark identification experiments, where the Sella (pituitary gland) pattern search was carried out using the same matching circuit. Since the 64-dimension PAP representation is essential for grayscale image recognition, the 64-dimension vector was divided into four

16-dimension vectors and the matching scores were measured separately and then summed up by off-chip calculation. The correct position was successfully identified both in the above-threshold (Fig. 8(b)) and the sub-threshold (Fig. 8(c)) regimes using the 14 learned vectors as templates. In the previous work [3], successful search was demonstrated by the computer simulation.

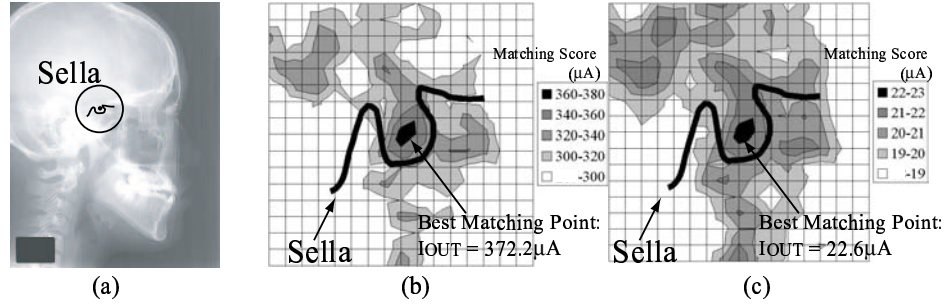

(a)  (b)  (c)

Figure 8: Matching results of Sella search using pyramid type with ABC=000: (a) input image; (b) above-threshold regime; (c) sub-threshold regime.

## 4.4  Separation of overlapping patterns

Suppose an unknown pattern is presented to the matching circuit. The pattern might consist of a single or multiple overlapping patterns. Let $\mathbf{X}$ represent the input vector and $\mathbf{W_{1st}}$ the winner (best matched) vector obtained by the matching circuit. Let the first matching trial be expressed as follows:

$$1\text{st trial:} \quad \mathbf{X} \xrightarrow{\quad matching \quad} \mathbf{W_{1st}}$$

Then, the residue vector ($\mathbf{X}$-$\mathbf{W_{1st}}$) is generated. The subtraction is perfomed in the vector space. When an element in the residue vector becomes negative, the value is set to 0. Such operation is easily implemented using the floating gate technique. Here, the residue was obtained by off-line calculation. If the input pattern is single, the residue vector is meaningless: only the leftover edge information remains in the residue vector. If the input consists of overlapping patterns, the edge information of other patterns remains. If the residue vector is very small, we can expect that the input is single. But in many cases, the residue vector is not so small due to the distortion in hand-written patterns. Thus, it is almost impossible to judge which is the case only from the magnitude of the residue vector. Therefore, we proceed to the second trial to find the second winner:

$$2\text{nd trial:} \quad \mathbf{X} - \mathbf{W_{1st}} \xrightarrow{\quad matching \quad} \mathbf{W_{2nd}}$$

With the same sequence, the second residue vector ($\mathbf{X}$-$\mathbf{W_{1st}}$-$\mathbf{W_{2nd}}$), the third ($\mathbf{X}$-$\mathbf{W_{1st}}$-$\mathbf{W_{2nd}}$-$\mathbf{W_{3rd}}$) and so forth are generated by repeating the winner subtraction after each trial. Then, new template vectors are generated such as $\mathbf{W_{1st}}$+$\mathbf{W_{2nd}}$, $\mathbf{W_{1st}}$+$\mathbf{W_{2nd}}$+$\mathbf{W_{3rd}}$, and so forth. If the input vector is that of a single pattern, the matching score is the highest at $\mathbf{W_{1st}}$ and the scores are lower at $\mathbf{W_{1st}}$+$\mathbf{W_{2nd}}$ and $\mathbf{W_{1st}}$+$\mathbf{W_{2nd}}$+$\mathbf{W_{3rd}}$. On the other hand, if the input vector is that of two overlapping patterns, the score is the highest at $\mathbf{W_{1st}}$+$\mathbf{W_{2nd}}$. This procedure can be terminated automatically when the new template composed of n overlapping patterns yields lower score than that of n-1 overlapping patterns. In this manner, we are able to

know how many patterns are overlapping and what patterns are overlapping without a priori knowledge. An example of separating multiple overlapping patterns is illustrated in Fig. 9.

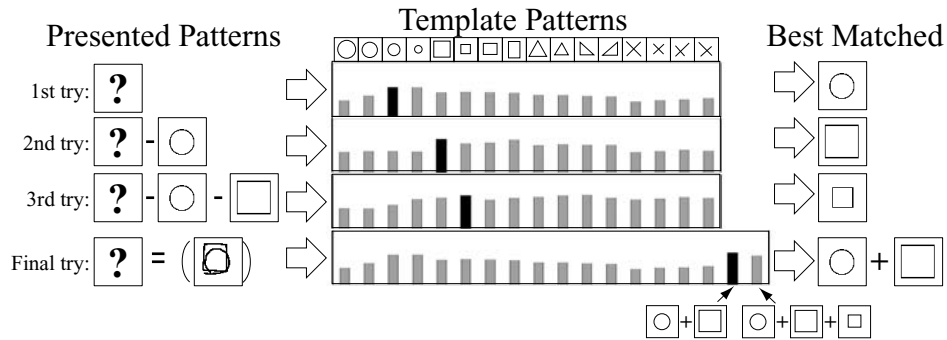

Figure 9: Experimental result illustrating the algorithm for separating overlapping patterns. The solid black bars indicate the winner locations.

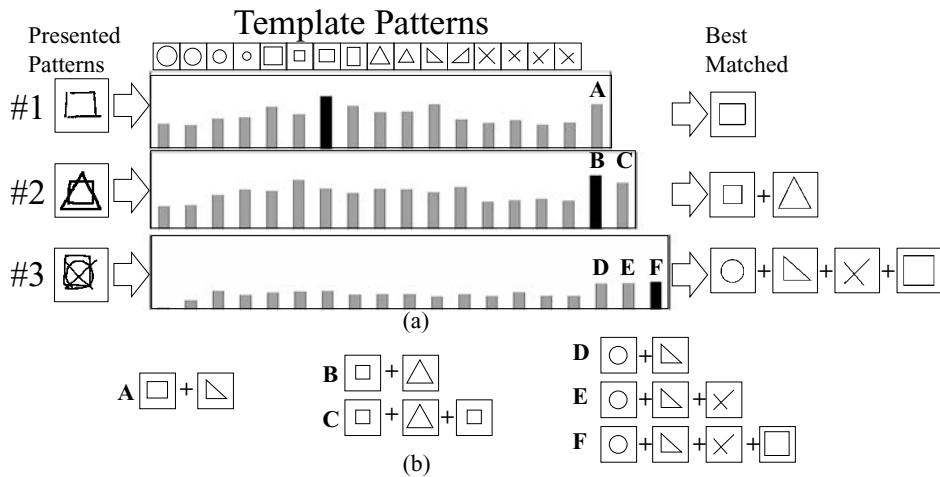

Figure 10: Measured results demonstrating separation of multiple overlapping patterns: (a) result of separation and classification (A~F are depicted in (b)); (b) newly created templates such as $W_{1st}+W_{2nd}, W_{1st}+W_{2nd}+W_{3rd}$, and so on.

Several other examples are shown in Fig. 10. Pattern #1 is correctly classified as a single rectangle by yielding the higher score for single template than that for $W_{1st}+W_{2nd}$. Pattern #3 consists of three overlapping patterns, but is erroneously recognized as four overlapping patterns. However, the result is not against human perception. When we look at pattern #3, a triangle is visible in the pattern. This mistake is quite similar to that made by humans.

## 5 Conclusions

A soft-pattern matching circuit has been demonstrated in conjunction with a robust image representation algorithm called PAP. The circuit has been successfully applied to hand-written pattern recognition and medical radiograph analysis. The recognition of overlapping patterns similar to human perception has been also experimentally demonstrated.

## Acknowledgments

Test circuits were fabricated in the VDEC program (The Univ. of Tokyo), in collaboration with Rohm Corp. and Toppan Printing Corp. The work is partially supported by the Ministry of Education, Science, Sports and Culture under the Grant-in-Aid for Scientific Research (No. 11305024) and by JST in the program of CREST.

## References

[1] G. T. Tuttle, S. Fallahi, and A.A. Abidi. (1993) An 8b CMOS Vector A/D Converter. *in ISSCC Tech. Digest, vol. 36*, pp. 38-39. IEEE Press.

[2] G. Cauwenberghs and V. Pedroni. (1995) A Charge-Based CMOS Parallel Analog Vector Quantizer. In G. Tesauro, D. S. Touretzky and T.K. Leen (eds.), *Advances in Neural Information Processing Systems 7*, pp. 779-786. Cambridge, MA: MIT Press.

[3] M. Yagi, M. Adachi, and T. Shibata. (2000) A Hardware-Friendly Soft-Computing Algorithm for Image Recognition. *X European Signal Processing Conf., Sept. 4-8, 2000 (EUSIPCO 2000), Vol. 2*, pp. 729-732, Tampere, Finland.

[4] M. Adachi and T. Shibata. (2001) Image Representation Algorithm Featuring Human Perception of Similarity for Hardware Recognition Systems. *In Proc. of the Int. Conf. on Artificial Intelligence (IC-AI'2001), Ed. by H. R. Arabnia, Vol. I*, 229-234 (CSREA Press, ISDBN: 1-892512-78-5), Las Vegas, Nevada, USA, June 25-28, 2001.

[5] T. Yamasaki and T. Shibata. (2001) An Analog Similarity Evaluation Circuit Featuring Variable Functional Forms. *In Proc. IEEE Int. Symp. Circuits Syst. (ISCAS 2001), Vol. 3*, pp. III-561-564, Sydney, Australia, May. 7-9, 2001.

[6] T. Yamasaki, K. Yamamoto and T. Shibata. (2001) Analog Pattern Classifier with Flexible Matching Circuitry Based on Principal-Axis-Projection Vector Representation. *In Proc. 27th European Solid-State Circ. Conf. (ESSCIRC 2001),* Ed. by F. Dielacher and H. Grunbacher, pp. 212-215 (Frontier Group), Villach, Austria, September 18-20, 2001.

[7] J. Anderson, J. C. Platt, and D. B. Kirk. (1993) An Analog VLSI Chip for Radial Basis Functions. In S. J. Hanson, J. D. Cowan, and C. L. Giles Eds., *Advances in Neural Information Processing Systems 5*, pp. 765-772., San Maetro, CA; Morgan Kaufmann.

[8] L. Theogarajan and L. A. Akers. (1996) A Multi-Dimentional Analog Gaussian Radial Basis Circuit. *In Proc. IEEE Int. Symp. Circuits Syst. (ISCAS '96)*, *Vol. 3*, pp. III-543-546 Atlanta, GA, USA, May, 1996.

[9] L. Theogarajan and L. A. Akers. (1997) A scalable low voltage analog Gaussian radial basis circuit. *IEEE Trans. on Circuits and Systems II, Volume 44, No. 11*, pp. 977-979, 1997.

[10] K. Ito, M. Ogawa and T. Shibata. (2001) A High-Performance Time-Domain Winner-Take-All Circuit Employing OR-Tree Architecture. *In Proc. 2001 Int. Conf. on Solid State Devices and Materials (SSDM 2001)*, pp. 94-95, Tokyo, Japan, Sep. 26-28, 2001.
